# Necessary Intransitive Likelihood-Ratio Classifiers

**Gang Ji** *and* **Jeff Bilmes**
SSLI-Lab, Department of Electrical Engineering
University of Washington
Seattle, WA 98195-2500
{*gang,bilmes*}*@ee.washington.edu*

## Abstract

In pattern classification tasks, errors are introduced because of differences between the true model and the one obtained via model estimation. Using likelihood-ratio based classification, it is possible to correct for this discrepancy by finding class-pair specific terms to adjust the likelihood ratio directly, and that can make class-pair preference relationships intransitive. In this work, we introduce new methodology that makes *necessary* corrections to the likelihood ratio, specifically those that are necessary to achieve perfect classification (but not perfect likelihood-ratio correction which can be overkill). The new corrections, while weaker than previously reported such adjustments, are analytically challenging since they involve discontinuous functions, therefore requiring several approximations. We test a number of these new schemes on an isolated-word speech recognition task as well as on the UCI machine learning data sets. Results show that by using the bias terms calculated in this new way, classification accuracy can substantially improve over both the baseline and over our previous results.

## 1 Introduction

Statistical pattern recognition is often based on Bayes decision theory [4], which aims to achieve minimum error rate classification. In previous work [2], we observed that multi-class Bayes classification can be viewed as a tournament style game, where the winner between players is decided using log likelihood ratios. Supposing the classes (players) are $\{c_1, c_2, \cdots, c_M\}$, and the observation (game) is $x$, the winner of each pair of classes is determined, with the assumption of equal priors, by the sign of the log likelihood ratio $L_{ij}(x) = \ln \frac{P(x|c_i)}{P(x|c_j)}$, in which case if $L_{ij} > 0$ class $c_i$ wins and otherwise class $c_j$ wins. A practical game strategy can be obtained by fixing a comparison order, $\{i_1, i_2, \cdots, i_M\}$, as a permutation of $\{1, 2, \cdots, M\}$, where class $c_{i_1}$ plays with class $c_{i_2}$, the winner plays with class $c_{i_3}$, and so on until a final winner is ultimately found. This yields a transitive game [8] — assuming no ties, the ultimate winner is identical regardless of the comparison order.

To perform these procedures optimally, correct likelihood ratios are needed, which requires correct probabilistic models and sufficient training data. This is never the case given a fi-

nite amount of training data or the wrong model family, typical in practice. In previous work [2], we introduced a method to correct for the difference between the true and an approximate log likelihood ratio. In this work, we improve upon the correction method by using an expression that can still lead to perfect correction, but is weaker than what we used before. We show that this new condition can achieve a significant improvement over baseline results, both on a medium vocabulary isolated-word automatic speech recognition task and on the UCI machine learning data sets. The paper is organized as follows: Section 2 describes the general scheme and describes past work. Section 3 discusses the weaker correction condition, and its approximations. Section 4 provides various experimental results on an isolated-word speech recognition task. Section 5 contains the experimental results on the UCI data. Finally, Section 6 concludes.

## 2  Background

A common problem in many probabilistic machine learning settings is the lack of a correct statistical model. In a generative pattern classification setting, this occurs because only an estimated quantity $\hat{P}(x|c)$[1] of a distribution is available, rather than the true class-conditional model $P(x|c)$. In the likelihood ratio decision scheme described above, only an imperfect log likelihood ratio, $\hat{L}_{ij}(x) = \ln(\hat{P}(x|c_i)/\hat{P}(x|c_j))$, is available for decision making rather than the true log likelihood ratio $L_{ij}(x)$.

One approach to correct for this inaccuracy is to use richer class conditional likelihoods, more complicated parametric forms of $L_{ij}(x)$ itself, and/or more training data. In previous work [2], we proposed a different approach that requires no change in generative models, no increase in free parameters, and no additional training data but still yields improved accuracy. The key idea is to compensate for the difference between $L_{ij}(x)$ and $\hat{L}_{ij}(x)$ using a *bias*[2] term $\alpha_{ij}(x)$ computed from test data such that:

$$L_{ij}(x) - \alpha_{ij}(x) = \hat{L}_{ij}(x). \tag{1}$$

If it is assumed that a single bias term is used for all data, so that $\alpha_{ij}(x) = \alpha_{ij}$, we found that the best $\alpha_{ij}$ is as follows:

$$\alpha_{ij} = \frac{1}{2}\left(D(i\|j) - D(j\|i)\right) - \frac{1}{2}\left(\hat{D}(i\|j) - \hat{D}(j\|i)\right), \tag{2}$$

where $D(i\|j) = E_{P(x|c_i)} \ln L_{ij}(x)$ is the Kullback-Leibler (KL) divergence [3] between $P(x|c_i)$ and $P(x|c_j)$ and $\hat{D}(i\|j) = E_{P(x|c_i)}\hat{L}_{ij}(x)$ is its estimation. Under the assumption (referred to as *assumption A* in Section 3.1) of symmetric KL-divergence for the true model (e.g., equal covariance matrices in the Gaussian case), the bias term can be solved explicitly as

$$\alpha_{ij} = -\frac{1}{2}\left(\hat{D}(i\|j) - \hat{D}(j\|i)\right). \tag{3}$$

We saw how the augmented likelihood ratio $S_{ij}(x) = \hat{L}_{ij}(x) + \alpha_{ij}$ can lead to an intransitive game [8, 13], since $S_{ij}(x)$ can specify intransitive preferences amongst the set $\{1, 2, \cdots, M\}$. We therefore investigated a number of intransitive game playing strategies. Moreover, we observed that if the correction was optimal, the true likelihood ratios would be obtained which are clearly transitive. We therefore hypothesized and experimentally verified that the existence of intransitivity was a good indicator of the occurrence of a classification error.

This general approach can be improved upon in several ways. First, better intransitive strategies can be developed (for detecting, tolerating, and utilizing the intransitivity of a

classifier); second, the assumption of symmetric KL-divergence could be relaxed; and third, the above criterion is stricter than required to obtain perfect correction. In this work, we advance on the latter two of the above three possible avenues for improvement.

## 3 Necessary Intransitive Scheme

An $\alpha_{ij}(x)$ that solves Equation 1 is a sufficient condition for a perfect correction of the estimated likelihood ratio since given such a quantity, the true likelihood ratio would be attainable. This condition, however, is stricter than required because it is only the sign of the likelihood ratio that is needed to decide the winning class. We therefore should ask for a condition that corrects only for the discrepancy in sign between the true and estimated ratio, i.e., we want to find a function $\alpha_{ij}(x)$ that minimizes

$$J[\alpha_{ij}] = \int_{\mathbb{R}^n} \left\{ \mathrm{sgn}\left[L_{ij}(x) - \alpha_{ij}(x)\right] - \mathrm{sgn}\hat{L}_{ij}(x) \right\}^2 \cdot P_{ij}(x)\, dx.$$

Clearly the $\alpha_{ij}(x)$ that minimizes $J[\alpha_{ij}]$ is the one such that

$$\mathrm{sgn}\left[L_{ij}(x) - \alpha_{ij}(x)\right] = \mathrm{sgn}\hat{L}_{ij}(x), \qquad \forall x \in \mathrm{supp}P_{ij} = \overline{\{x : P_{ij}(x) \neq 0\}}. \quad (4)$$

As can be seen, this condition is weaker than Equation 1, weaker in the sense that any solution to Equation 1 solves Equation 4 but not vice versa. Note also that Equation 4 provides *necessary* conditions for an additive bias term to achieve perfect correction, since any such correction must achieve parity in the sign. Therefore, it might make it simpler to find a better bias term since Equation 4 (and therefore, set of possible $\alpha$ values) is less constrained. As will be seen, however, analysis of this weaker condition is more difficult. In the following sections, therefore we introduce several approximations to this condition.

Note that as in previous work, we henceforth assume $\alpha_{ij}(x) = \alpha_{ij}$ is a constant. In this case, the equation providing the best $\alpha_{ij}$ values is:

$$E_{P_{ij}} \left\{ \mathrm{sgn}\left[L_{ij}(x) - \alpha_{ij}\right] \right\} = E_{P_{ij}} \left\{ \mathrm{sgn}\hat{L}_{ij}(x) \right\}. \quad (5)$$

### 3.1 The difficulty with the sign function

The main problem in trying to solve for $\alpha_{ij}$ in Equation 5 is the existence of a discontinuous function. In this section, therefore, we work towards obtaining an analytically tractable approximation. The $\{-1, 0, 1\}$-valued sign function $\mathrm{sgn}(z)$ is defined as $2u(z) - 1$, where $u(z)$ is the Heaviside step function. We obtain an approximation via a Taylor expansion as follows:

$$\mathrm{sgn}(z + \epsilon) = \mathrm{sgn}(z) + \epsilon\,\mathrm{sgn}'(z) + o(\epsilon) = \mathrm{sgn}(z) + 2\epsilon\delta(z) + o(\epsilon), \quad (6)$$

where $\delta(z)$ is the Dirac delta function [7]. It can be defined as the derivative of the Heaviside step function $u'(z) = \delta(z)$, and it satisfies the sifting property $\int_{\mathbb{R}} f(z)\delta(z - z_0) = f(z_0)$. Therefore, it follows that [6, page 263]

$$\int_{\mathbb{R}^n} f(z)\delta[g(z)]\, dz = \int_{Z_g} \frac{f(z)}{|\nabla g(z)|} \cdot d\mu,$$

where $\nabla g$ is the gradient of $g$ and $Z_g = \{z \in \mathbb{R}^n : g(z) = 0\}$ is the zero set of $g$ with Lebesgue measure $\mu$ [12].

Of course, the Taylor expansion is valid only for a differentiable function, otherwise the error terms can be arbitrarily large. If, however, we find and use a suitable continuous and

differentiable approximation rather than the discrete sign function, the above expansion becomes more appropriate. There exists a trade-off, however, between the quality of the sign function approximation (a better sign function should yield a better approximation in Equation 4) and the error caused by the $o(\epsilon)$ term in Equation 6 (a better sign function approximation will have a greater error when the higher-order Taylor terms are dropped). We therefore expect that ideally there will exist an optimal balance between the two. The shifted sigmoid with free parameter $\beta$ (defined and used below) allows us to easily explore this trade-off simply by varying $\beta$.

Retaining the first-order Taylor term, and applying this to the left side of Equation 5,

$$E_{P_{ij}}\operatorname{sgn}\left[L_{ij}(x) - \alpha_{ij}\right] \approx E_{P_{ij}}\operatorname{sgn}L_{ij}(x) - 2E_{P_{ij}}\alpha_{ij}\delta\left[L_{ij}(x)\right].$$

The distribution under which the expectation in Equation 5 is taken can also influence our results. If it is known that the true class of $x$ is always $c_i$, the $c_i$-conditional distribution should be used, i.e., $P_{ij}(x) = P(x|c_i)$, yielding a class-conditional correction term $\alpha_{ij}^{(i)}$, and a class-conditional likelihood-ratio correction $S_{ij}^{(i)}(x) = \hat{L}_{ij}(x) + \alpha_{ij}^{(i)}$. The symmetric case arises when $x$ is of class $c_j$. If, on the other hand, neither $c_i$ nor $c_j$ is the true classes (i.e., $x$ is sampled from some other class-conditional distribution, say $P(x|c_k)$, $k \neq i, j$), it does not matter which distribution for $P_{ij}(x)$ is used since, for a given comparison order in a game playing strategy, the current winner will ultimately play using the true class distribution $P(x|c_k)$ of $x$ (when one of $i$ or $j$ will equal $k$). It is therefore valid to consider only the case when either $x$ is of class $c_i$ (we denote this event by $C_i(x)$) or when $x$ is of class $c_j$ (event $C_j(x)$). Note that these two events are disjoint.

In practice, however, we do not know which of the two events is correct. The ideal choice in either case can be expressed using indicators as follows:

$$A_{ij}(x) = \alpha_{ij}^{(i)}\mathbf{1}_{\{C_i(x)\}} + \alpha_{ij}^{(j)}\mathbf{1}_{\{C_j(x)\}}.$$

Taking the expected value of $A_{ij}(X)$ with respect to $p(x|C_i(x) \vee C_j(x))$ yields

$$\alpha_{ij} = E_{p(x|C_i(x)\vee C_j(x))}[A_{ij}(X)] = \frac{\alpha_{ij}^{(i)}P(c_i) + \alpha_{ij}^{(j)}P(c_j)}{P(c_i) + P(c_j)}.$$

This results in a single likelihood correction $S_{ij}(x) = \hat{L}_{ij}(x) + \alpha_{ij}$ that is obtained simply by integrating in Equation 5 with respect to the average distribution over class $c_i$ and $c_j$, i.e.,

$$P_{ij}(x) \triangleq p(x|C_i(x) \vee C_j(x)) = \frac{P(c_i)P(x|c_i) + P(c_j)P(x|c_j)}{P(c_i) + P(c_j)}.$$

With these assumptions, and supposing the zero set $Z_{L_{ij}} = \{x \in \mathbb{R}^n : P(x|c_i) = P(x|c_j)\}$ of $L_{ij}(x)$ is Lebesgue measurable with measure $\mu$, we get:

$$\int_{\mathbb{R}^n}\left\{\operatorname{sgn}L_{ij}(x) - 2\alpha_{ij}\delta\left[L_{ij}(x)\right]\right\}P_{ij}(x)\,dx = \int_{\mathbb{R}^n}\operatorname{sgn}L_{ij}(x)P_{ij}(x)\,dx - 2\Psi(P_i, P_j)\alpha_{ij},$$

where

$$\Psi(P_i, P_j) = \int_{\mathbb{R}^n}P_{ij}(x)\delta\left[L_{ij}(x)\right]\,dx = \int_{Z_{L_{ij}}}\frac{P_{ij}(x)}{|\nabla L_{ij}(x)|}\cdot d\mu. \tag{7}$$

Therefore,

$$\alpha_{ij} = \frac{1}{\Psi(P_i, P_j)}\int_{\mathbb{R}^n}\left[\frac{\operatorname{sgn}L_{ij}(x) - \operatorname{sgn}\hat{L}_{ij}(x)}{2}\right]P_{ij}(x)\,dx.$$

As can be seen, $\alpha_{ij}$ is composed of two factors, the integral and the $1/\Psi(P_i, P_j)$ factor. The integral is bounded between -1 and 1 and determines the direction of the correction. When $L_{ij}(x)$ and $\hat{L}_{ij}(x)$ always agree, the integral is zero and there is no correction. The correction favors $i$ when $\alpha_{ij}$ is positive. This occurs when $L_{ij}$ is positive and $\hat{L}_{ij}$ is negative more often than $L_{ij}$ is negative and $\hat{L}_{ij}$ is positive, a situation improved upon by giving $i$ "help." Similarly, when $\alpha_{ij}$ is negative, the correction biases towards $j$.

The maximum amount of absolute likelihood correction possible is determined by the (always positive) $1/\Psi(P_i, P_j)$ factor. This is affected by two quantities, the mass around and the log-likelihood ratio gradient at the decision boundary. Low mass at the decision boundary increases the maximum possible correction because any errors in the integral factor are being de-weighted. High gradient at the decision boundary also increases the maximum possible correction because any decision boundary deviation causes a higher change in likelihood ratio than if the gradient was low. Since we are correcting the likelihood ratio directly, this needs to be reflected in $\alpha_{ij}$.

When $P(x|c_i)$ and $P(x|c_j)$ are multivariate Gaussians with means $\mu_i$ and $\mu_j$, identical covariance matrices $\Sigma$, and equal priors, this becomes:

$$\Psi(P_i, P_j) = \frac{e^{-\frac{1}{8}(\mu_i - \mu_j)^T \Sigma^{-1}(\mu_i - \mu_j)}}{\sqrt{2\pi(\mu_i - \mu_j)^T \Sigma^{-1}(\mu_i - \mu_j)}}$$

As the means diverge from each other, both the mass at the decision boundary decreases and the likelihood-ratio gradient increases, thereby increasing the maximum amount of correction.

Unfortunately, it is quite difficult to explicitly evaluate $\Psi(P_i, P_j)$ without knowing the true probability distributions. In this initial work, therefore, our investigations simplify by only computing the direction and not the magnitude of the correction. As will be seen, this assumption yields a likelihood-ratio adjustment that is similar in form to our previous KL-divergence based adjustment. More practically, the assumption significantly simplifies the derivation and still yields reasonable empirical results. Under this assumption, expression for $\alpha_{ij}$ becomes:

$$\alpha_{ij} = \frac{1}{2}E_{P_{ij}(x)}[\mathrm{sgn}L_{ij}(x)] - \frac{1}{2}E_{P_{ij}(x)}[\mathrm{sgn}\hat{L}_{ij}(x)]. \tag{8}$$

The left term on the right of the equality is quite similar to the left difference on the right of the equality in the KL-divergence case (Equation 2). Again, because we have no information about the true class conditional models, we assume the left term in Equation 8 to be zero (denote this as assumption $B$). Comparing this with the corresponding assumption for the KL-divergence case (assumption $A$, Equations 2 and 3), it can be shown that 1) they are not identical in general, and 2) in the Gaussian case, $A$ implies $B$ but not vice versa, meaning $B$ is weaker than $A$.

Under assumption $B$, an expression for the resulting $\alpha_{ij}$ can be derived using the weak law of large numbers yielding:

$$\alpha_{ij} \approx \frac{1}{2(N_i + N_j)}\left(\sum_{x \in C_i} \mathrm{sgn}\ln \frac{\hat{P}(x|c_j)}{\hat{P}(x|c_i)} - \sum_{x \in C_j} \mathrm{sgn}\ln \frac{\hat{P}(x|c_i)}{\hat{P}(x|c_j)}\right), \tag{9}$$

where $x \in C_i$ and $x \in C_j$ correspond to the samples as they are classified in a previous recognition pass; $N_i$ and $N_j$ are number of samples from model $c_i$ and $c_j$ respectively. One can immediately see the similarity between this equation and the one using KLD [2].

Like in [2], since the true classes are unknown, we perform a previous classification pass (e.g., using the original likelihood ratios) to get estimates and use these in Equation 9.

Note that there are three potential sources of error in the analysis above. The first is the $\Psi(P_i, P_j)$ factor that we neglected. The second is assumption $B$, that (since weaker) can be less severe than in the corresponding KL-divergence case. The third is the error due to the discontinuity of the sign function. To address the third problem, rather than using the sign function in Equation 9, we can approximate it with a continuous differential function with the goal of balancing the trade-off mentioned above. There are a number of possible sign-function approximations, including hyperbolic and arc tangent, and shifted sigmoid function, the latter of which is the most flexible because of its free parameter $\beta$.[3]

Specifically, the sigmoid function has the form $f(z) = \frac{1}{1+e^{-\beta z}}$, where the free parameter $\beta$ (an inverse temperature) determines how well the curve will approximate the discontinuous function. Using the sigmoid function, we can approximate the sign function as $\text{sgn} z \approx \frac{2}{1+e^{-\beta z}} - 1$. Note that the approximation improves as $\beta$ increases. Hence,

$$\alpha_{ij} \approx \frac{1}{2(N_i + N_j)} \left[ \sum_{x \in c_i} \left( 1 - \frac{2}{1 + e^{\beta \hat{L}_{ji}(x)}} \right) - \sum_{x \in c_j} \left( 1 - \frac{2}{1 + e^{\beta \hat{L}_{ij}(x)}} \right) \right]. \quad (10)$$

## 4  Speech Recognition Evaluation

As in previous work [2], we implemented this technique on NYNEX PHONEBOOK [10, 1], a medium vocabulary isolated-word speech corpus. Gaussian mixture hidden Markov models (HMMs) produced probability scores $\hat{P}(x|c_i)$ where here $x$ is a matrix of feature values (one dimension as MFCC features and the other as time frames), and $c_i$ is a word identity. The HMMs use four hidden states per phone, and 12 Gaussian mixtures per state (standard for this task [10]). This yields approximately 200k free model parameters in total.

In our experiments, the steps are: 1) calculate $\hat{P}(x|c_i)$ using full inference (no Viterbi approximation) for each test case and for each word; 2) classify the test examples using just the log likelihood ratios $\hat{L}_{ij} = \ln \hat{P}(x|c_i)/\hat{P}(x|c_j)$; 3) using the hypothesized (and error-full) class labels, calculate the test-set bias term using one of the techniques described above; and 4) classify again using the augmented likelihood ratio $S_{ij} = \hat{L}_{ij} + \alpha_{ij}$. Since the procedure is no longer transitive, we run 1000 random tournament-style games (as in [2]) and choose the most frequent winner as the ultimate winner.

Table 1: Word error rates % on speech data with various sign approximations.

| SIZE | ORIG | SIGN | TANH | ATAN | SIG(.1) | SIG(1) | SIG(10) | SIG(100) | SIG(200) | SIG(400) | KLD[2] |
|------|------|------|------|------|---------|--------|---------|----------|----------|----------|--------|
| 75   | 2.34 | 1.76 | 1.76 | 1.76 | 1.82    | 1.76   | 1.56    | 1.57     | 1.33     | 1.34     | 1.91   |
| 150  | 3.31 | 2.83 | 2.84 | 2.83 | 2.65    | 2.83   | 2.65    | 2.47     | 2.68     | 2.43     | 2.72   |
| 300  | 5.23 | 4.75 | 4.75 | 4.70 | 4.74    | 4.75   | 4.29    | 3.95     | 4.34     | 4.34     | 4.29   |
| 600  | 7.39 | 6.64 | 6.61 | 6.60 | 6.66    | 6.64   | 6.04    | 5.70     | 6.74     | 6.74     | 5.91   |

The results are shown in Table 1, where the first column gives the test-set vocabulary size (number of different classes). The second column shows the baseline word error rates (WERs) using only $\hat{L}_{ij}$. The remaining columns are the bias-corrected results with various sign approximations, namely sign (Equation 9), hyperbolic and arc tangent, and the shifted sigmoid with various $\beta$ values (thus allowing us to investigate the trade-off mentioned in Section 3.1). From the results we can see that larger-$\beta$ sigmoid is usually better, with overall performance increasing with $\beta$. This is because with large $\beta$, the shifted sigmoid curve better approximates the sign function. For $\beta = 100$, the results are even better than our previous KL-divergence (KLD) results reported in [2] (right-most column in the table). It can also been seen that when $\beta$ is greater than 100, the WERs are *not* consistently better. This indicates that the inaccuracies due to the Taylor error term start adversely affecting the results at around $\beta = 100$.

# 5 UCI Dataset Evaluation

Table 2: Error rates in % (and std where applicable) on the UCI data.

| data | NN baseline | KLD | sign | sig(10) | NB baseline | KLD | sign | sig(10) |
|------|-------------|-----|------|---------|-------------|-----|------|---------|
| australian | 16.75(3.51) | 16.33(3.66) | 16.17(3.63) | 16.32(3.75) | 14.89(1.97) | 14.29(2.45) | 14.76(2.45) | 14.76(2.37) |
| breast | 2.94(1.16) | 2.62(1.15) | 2.63(1.15) | 2.65(1.15) | 2.45(1.93) | 2.29(2.02) | 2.13(2.07) | 1.86(2.07) |
| chess | 0.56 | 0.46 | 0.47 | 0.37 | 12.66 | 12.76 | 13.04 | 12.85 |
| cleve | 25.67(3.40) | 24.35(2.82) | 24.01(2.27) | 24.01(3.94) | 17.91(2.37) | 15.55(1.81) | 15.22(1.82) | 16.22(2.61) |
| corral | 2.44(1.26) | 1.82(1.16) | 1.19(1.16) | 1.19(1.16) | 12.77(3.66) | 9.57(2.12) | 9.57(2.62) | 12.05(4.80) |
| crx | 17.41(3.18) | 17.25(2.67) | 17.11(2.91) | 17.26(3.00) | 15.05(3.67) | 14.02(3.91) | 13.06(3.67) | 15.05(3.67) |
| diabetes | 28.04(3.08) | 26.88(3.56) | 27.41(4.13) | 27.18(1.98) | 25.71(2.13) | 24.79(2.68) | 24.24(3.49) | 24.66(2.59) |
| flare | 20.98(2.26) | 19.37(2.16) | 18.29(2.25) | 18.46(1.85) | 20.24(2.31) | 19.55(2.63) | 18.70(1.87) | 16.64(2.34) |
| german | 29.96(3.49) | 28.54(3.45) | 28.82(2.53) | 28.25(3.71) | 24.58(2.57) | 26.55(1.88) | 24.79(2.30) | 24.25(2.50) |
| glass | 42.16(2.06) | 39.63(1.76) | 41.92(1.92) | 40.95(2.00) | 44.12(7.96) | 42.24(8.64) | 42.06(9.22) | 42.28(7.93) |
| glass2 | 28.82(2.57) | 26.23(2.61) | 26.95(2.65) | 26.23(2.57) | 22.36(9.01) | 21.15(9.25) | 21.77(9.25) | 22.36(9.01) |
| heart | 21.83(3.77) | 21.48(4.26) | 21.19(4.52) | 21.09(4.23) | 15.50(6.01) | 15.11(5.34) | 15.11(5.72) | 15.11(6.01) |
| hepatitis | 19.46(7.10) | 16.10(6.13) | 17.16(6.92) | 15.82(6.94) | 16.18(5.92) | 18.29(5.96) | 18.04(5.92) | 15.45(4.56) |
| iris | 8.13(1.60) | 6.84(1.44) | 6.26(1.47) | 6.84(1.44) | 6.99(1.78) | 6.99(1.78) | 6.99(1.78) | 6.99(1.78) |
| letter | 38.66 | 34.66 | 37.10 | 37.00 | 30.68 | 30.88 | 30.48 | 30.64 |
| lymphography | 24.46(4.86) | 23.81(4.57) | 23.29(4.52) | 23.29(4.86) | 16.62(8.64) | 18.27(9.25) | 17.34(8.91) | 15.31(8.91) |
| mofn-3-7-10 | 0 | 0 | 0 | 0 | 8.59 | 4.57 | 1.56 | 3.42 |
| pima | 25.96(2.01) | 25.22(2.95) | 24.82(2.87) | 25.96(2.19) | 25.71(2.13) | 24.79(2.68) | 24.24(3.49) | 24.66(2.59) |
| satimage | 15.80 | 14.25 | 14.40 | 14.25 | 19.15 | 19.35 | 19.25 | 18.70 |
| segment | 7.53 | 7.40 | 7.27 | 7.53 | 12.21 | 11.73 | 11.82 | 12.21 |
| shuttle-small | 0.87 | 0.77 | 0.87 | 0.77 | 1.40 | 1.41 | 1.50 | 1.50 |
| soybean-large | 8.47(1.31) | 8.29(1.39) | 7.18(1.08) | 8.47(1.31) | 8.71(2.70) | 9.13(2.60) | 8.35(2.65) | 8.37(2.70) |
| vehicle | 28.39(4.68) | 28.15(4.62) | 27.70(4.44) | 28.39(4.75) | 38.92(4.47) | 38.59(5.05) | 38.79(4.46) | 37.84(4.43) |
| vote | 7.40(2.22) | 6.94(1.77) | 6.94(1.77) | 7.17(2.05) | 9.91(1.72) | 9.68(2.49) | 9.68(1.72) | 9.68(1.72) |
| waveform-21 | 26.21 | 26.17 | 26.12 | 26.14 | 21.45 | 21.11 | 20.15 | 21.40 |

In order to show that our methodology is general beyond isolated-word speech recognition, we also evaluated this technique on the entire UCI machine learning repository [9]. In our experiments, baseline classifiers are built using one of: 1) the Matlab neural network (NN) toolbox with feed-forward 3-layer perceptrons having different number of hidden units and training epochs (optimized over a large set to achieve the best possible baseline for each test case), and trained using the Levenberg-Marquardt algorithm [11], or 2) the MLC++ toolbox to produce naïve Bayes (NB) classifiers that have been smoothed using Dirichlet priors. In each case (i.e., NN or NB), we augmented the resulting likelihood ratios with bias correction terms thereby evaluating our technique using quite different forms of baseline classifiers. Unlike the above, with these data sets we have only tried one random tournament game to decide the winner so far.

For the NN results, hidden units use logistic sigmoid, and output units use a soft-max function, making the network outputs interpretable as posterior probabilities $P(c|x)$, where $x$ is the sample and $c$ is the class. While our bias correction described above is in terms of likelihoods ratios $L_{ij}(x)$, posteriors can be used as well if the posteriors are divided by the priors giving the relation $P(c|x)/P(c) = P(x|c)/p(x)$ (i.e., scaled likelihoods) which produces the standard $L_{ij}(x)$ values when used in a likelihood ratio .

As was done in [5], for the small data sets the experimental results use 5-fold cross-validation using randomly selected chunks — results show mean and standard deviation (std) in parentheses. For the larger data sets, we use the same held out training/test sets as in [5] (so std is not shown). The experimental procedure is similar to that described in Section 4, except that scaled likelihoods are used for the NN baselines. Again, first-pass error-full test-set hypothesized answers are used to compute the bias corrections.

Table 5 shows our results for both the NN (columns 2—5) and NB (columns 6—9) baseline classifiers. Within each baseline group, the first column shows the baseline accuracy (with the 5-fold standard derivations when the data set is small). The second column shows results using KL-divergence based bias corrections — these are the first published KLD results on the UCI data. The third column shows results with sign-based correction (Equation 9), and the forth column shows the sigmoid ($\beta = 10$) case (Equation 10).

While not the point of this paper, one immediately sees that the NB baseline results are often better than the NN baseline results (15 out of 25 times). Using the NN as a baseline,

the table shows that the KLD results are almost always better than the baseline 24 times (out of 25). Also, the sign correction is better than the baseline 23 out of 25 times, and the sigmoid(10) results are better 20 times. Also (not shown in the table), we found that $\beta = 10$ is slightly better than $\beta = 1$ but there is no advantage using $\beta = 100$. These results therefore show that the NN KLD correction typically beats the sign and sigmoid correction, possibly owing to the error in the Taylor approximation. Using the NB classifier as the baseline, however, shows not only improved baseline results in general but also that the sigmoid(10) improves more often. Specifically, the KLD results are better than the baseline 16 times, sign is better than the baseline 18 times, and sigmoid(10) beats the baseline 19 times, suggesting that sigmoid(10) typically wins over the KLD case.

## 6  Discussion

We have introduced a new necessary intransitive likelihood ratio classifier. This was done by using sign-based corrections to likelihood ratios and by using continuous differentiable approximations of the sign function in order to be able to vary the inherent trade-off between sign-function approximation accuracy and Taylor error. We have applied these techniques to both a speech recognition corpus and the UCI data sets, as well as applying previous KL-divergence based corrections to the latter data. Results on the UCI data sets confirm that our techniques reasonably generalize to data sets other than speech recognition. This suggests that the framework could be applied to other machine learning tasks.

This work was supported in part by NSF grant IIS-0093430 and IIS-0121396.

## Footnotes

[1]In this paper, we use "hatted" letters to describe estimated quantities.

[2]Note that by *bias*, we do not mean standard parameter bias in statistical parameter estimation.

[3]Note that the other soft sign functions can also be defined to utilize a $\beta$ smoothness parameter.

## References

[1] Jeff Bilmes. Burried Markov models for speech recognition. In *IEEE Intl. Conf. on Acoustics, Speech, and Signal Processing*, March 1999.

[2] Jeff Bilmes, Gang Ji, and M. Meilă. Intransitive likeilhood-ratio classifiers. In *Neural Information Processing Systems: Natural and Synthetic*, December 2001.

[3] T. M. Cover and J. A. Thomas. *Elements of Information Theory*. John Wiley and Sons, Inc., 1991.

[4] Richard O. Duda, Peter E. Hart, and David G. Stork. *Pattern Classification*. John Wiley and Sons, second edition, 2001.

[5] Nir Friedman, Dan Geiger, and Moises Goldszmidt. Bayesian network classifiers. *Machine Learning*, 29(2-3):131–163, 1997.

[6] D. S. Jones. *Generalised Functions*. McCraw-Hill Publishing Company Limited, 1966.

[7] J. Kevorkian. *Partial Differential Equations: Analytical Solution Techniques*. New York: Springer, 2000.

[8] R. Duncan Luce and Howard Raiffa. *Games and Decisions: Introduction and Critical Survey*. Dover, 1957.

[9] P. M. Murphy and D. W. Aha. *UCI Repository of Machine Learning Database*, 1995.

[10] J. Pitrelli, C. Fong, S. H. Wong, J. R. Spitz, and H. C. Lueng. PhoneBook: a phonetically-rich isolated-word telephone-speech database. In *IEEE Intl. Conf. on Acoustics, Speech, and Signal Processing*, 1995.

[11] W. H. Press, B. P. Flannery, S. A. Teukolsky, and W. T. Vetterling. *Numerical Recipes in C: The Art of Scientific Computing*. Cambridge University Press, Cambridge, England, second edition, 1992.

[12] M. M. Rao. *Measure Theory and Integration*. John Wiley and Sons, 1987.

[13] P. D. Straffin. *Game Theory and Strategy*. The Mathematical Association of America, 1993.
